# Sparse Estimation Using General Likelihoods and Non-Factorial Priors

**David Wipf and Srikantan Nagarajan,** [*]
Biomagnetic Imaging Lab, UC San Francisco
{david.wipf, sri}@mrsc.ucsf.edu

## Abstract

Finding maximally sparse representations from overcomplete feature dictionaries frequently involves minimizing a cost function composed of a likelihood (or data fit) term and a prior (or penalty function) that favors sparsity. While typically the prior is factorial, here we examine non-factorial alternatives that have a number of desirable properties relevant to sparse estimation and are easily implemented using an efficient and globally-convergent, reweighted $\ell_1$-norm minimization procedure. The first method under consideration arises from the sparse Bayesian learning (SBL) framework. Although based on a highly non-convex underlying cost function, in the context of canonical sparse estimation problems, we prove uniform superiority of this method over the Lasso in that, (i) it can never do worse, and (ii) for any dictionary and sparsity profile, there will always exist cases where it does better. These results challenge the prevailing reliance on strictly convex penalty functions for finding sparse solutions. We then derive a new non-factorial variant with similar properties that exhibits further performance improvements in some empirical tests. For both of these methods, as well as traditional factorial analogs, we demonstrate the effectiveness of reweighted $\ell_1$-norm algorithms in handling more general sparse estimation problems involving classification, group feature selection, and non-negativity constraints. As a byproduct of this development, a rigorous reformulation of sparse Bayesian classification (e.g., the relevance vector machine) is derived that, unlike the original, involves no approximation steps and descends a well-defined objective function.

## 1  Introduction

With the advent of compressive sensing and other related applications, there has been growing interest in finding sparse signal representations from redundant dictionaries [3, 5]. The canonical form of this problem is given by,

$$\min_{\boldsymbol{x}} \|\boldsymbol{x}\|_0, \qquad \text{s.t. } \boldsymbol{y} = \Phi\boldsymbol{x}, \tag{1}$$

where $\Phi \in \mathbb{R}^{n \times m}$ is a matrix whose columns $\boldsymbol{\phi}_i$ represent an *overcomplete* or redundant basis (i.e., $\text{rank}(\Phi) = n$ and $m > n$), $\boldsymbol{x} \in \mathbb{R}^m$ is a vector of unknown coefficients to be learned, and $\boldsymbol{y}$ is the signal vector. The cost function being minimized represents the $\ell_0$ norm of $\boldsymbol{x}$ (i.e., a count of the number of nonzero elements in $\boldsymbol{x}$). If measurement noise or modeling errors are present, we instead solve the alternative problem

$$\min_{\boldsymbol{x}} \|\boldsymbol{y} - \Phi\boldsymbol{x}\|_2^2 + \lambda\|\boldsymbol{x}\|_0, \quad \lambda > 0, \tag{2}$$

noting that in the limit as $\lambda \to 0$, the two problems are equivalent (the limit must be taken outside of the minimization). From a Bayesian perspective, optimization of either problem can be viewed, after a $\exp[-(\cdot)]$ transformation, as a challenging MAP estimation task with a quadratic likelihood function and a prior that is both improper and discontinuous. Unfortunately, an exhaustive search for the optimal representation requires the solution of up to $\binom{m}{n}$ linear systems of size $n \times n$, a

---

[*]This research was supported by NIH grants R01DC04855 and R01DC006435.

prohibitively expensive procedure for even modest values of $m$ and $n$. Consequently, in practical situations there is a need for approximate methods that efficiently solve (1) or (2) with high probability. Moreover, we would ideally like these methods to generalize to other likelihood functions and priors for applications such as non-negative sparse coding, classification, and group variable selection.

One common strategy is to replace $\|\boldsymbol{x}\|_0$ with a more manageable penalty function $g(\boldsymbol{x})$ (or prior) that still favors sparsity. Typically this replacement is a concave, non-decreasing function of $|\boldsymbol{x}| \triangleq [|x_1|, \ldots, |x_m|]^T$. It is also generally assumed to be *factorial*, meaning $g(\boldsymbol{x}) = \sum_i g(x_i)$. Given this selection, a recent, very successful optimization technique involves iterative reweighted $\ell_1$ minimization, a process that produces more focal estimates with each passing iteration [3, 19]. To implement this procedure, at the $(k+1)$-th iteration we compute

$$\boldsymbol{x}^{(k+1)} \rightarrow \arg \min_{\boldsymbol{x}} \|\boldsymbol{y} - \Phi\boldsymbol{x}\|_2^2 + \lambda \sum_i w_i^{(k)} |x_i|, \qquad (3)$$

where $w_i^{(k)} \triangleq \partial g(x_i^{(k)})/\partial |x_i^{(k)}|$. As discussed in [6], these updates are guaranteed to converge to a local minimum of the underlying cost function by satisfying the conditions of the *Global Convergence Theorem* (see for example [24]). Moreover, empirical evidence from [3] suggests that generally only a few iterations, which can be readily computed using standard convex programming packages, are required. Note that a single iteration with unit weights is equivalent to the traditional Lasso estimator [14]. However, given an appropriate selection for $g(\cdot)$, e.g., $g(x_i) = \log(|x_i| + \alpha)$ with $\alpha > 0$, subsequent iterations have been shown to exhibit substantial improvements over the Lasso in approximating the solution of (1) or (2) [3].

While certainly successful in practice, there remain fundamental limitations as to what can be achieved using factorial penalties to approximate $\|\boldsymbol{x}\|_0$. Perhaps counterintuitively, it has been shown in [19] that by considering the wider class of *non-factorial* penalties, more effective surrogates for $\|\boldsymbol{x}\|_0$ can be obtained, potentially leading to better approximate solutions of either (1) or (2). In this paper we consider two non-factorial methods that rely on the same basic iterative reweighted $\ell_1$ minimization procedure outlined above. In Section 2, we briefly introduce the non-factorial penalty function first proposed in [19] (based on a dual-form interpretation of sparse Bayesian learning) and then derive a new iterative reweighted $\ell_1$ implementation that builds upon these ideas. We then demonstrate that this algorithm satisfies two desirable properties pertaining to problem (1): (i) each iteration can only improve the sparsity and, (ii) for any $\Phi$ and sparsity profile, there will always exist cases where performance improves over standard $\ell_1$ minimization, which represents the best convex approximation to (1). Together, these results imply that this reweighting scheme can never do worse than Lasso (assuming $w_i^{(0)} = 1, \forall i$), and that there will always be cases where improvement over Lasso is achieved. To a large extent, this removes much of the stigma commonly associated with using non-convex sparsity penalties. Later in Section 3, we derive a second promising non-factorial variant by starting with a plausible $\ell_1$ reweighting scheme and then working backwards to determine the form and properties of the underlying penalty function.

In general, iterative reweighted $\ell_1$ procedures of any kind are attractive for our purposes because they can easily be augmented to handle other likelihoods and priors, provided convexity of the update (3) is preserved (of course the overall cost function being minimized will be non-convex). For example, to address the extensions mentioned above, in Section 4 we explore adding constraints such as $x_i \geq 0$, replacing $|x_i|$ with a norm on groups of variables, and using a logistic instead of quadratic likelihood term for classification. The latter extension leads to a rigorous reformulation of sparse Bayesian classification (e.g., the relevance vector machine [15]) that, unlike the original, involves no approximation steps and descends a well-defined objective function. Finally, Section 5 contains empirical comparisons while Section 6 provides brief concluding remarks.

## 2 Non-Factorial Methods Based on Sparse Bayesian Learning

A particularly useful non-factorial penalty emerges from a dual-space view [19] of sparse Bayesian learning (SBL) [15], which is based on the notion of automatic relevance determination (ARD) [10]. SBL assumes a Gaussian likelihood function $p(\boldsymbol{y}|\boldsymbol{x}) = \mathcal{N}(\boldsymbol{y}; \Phi\boldsymbol{x}, \lambda I)$, consistent with the data fit term from (2). The basic ARD prior incorporated by SBL is $p(\boldsymbol{x}; \boldsymbol{\gamma}) = \mathcal{N}(\boldsymbol{x}; 0, \text{diag}[\boldsymbol{\gamma}])$, where $\boldsymbol{\gamma} \in \mathbb{R}_+^m$ is a vector of $m$ non-negative hyperparameters governing the prior variance of each

unknown coefficient. These hyperparameters are estimated from the data by first marginalizing over the coefficients $\boldsymbol{x}$ and then performing what is commonly referred to as evidence maximization or type-II maximum likelihood [10, 15]. Mathematically, this is equivalent to minimizing

$$\mathcal{L}(\boldsymbol{\gamma}) \triangleq -\log \int p(\boldsymbol{y}|\boldsymbol{x})p(\boldsymbol{x};\boldsymbol{\gamma})d\boldsymbol{x} = -\log p(\boldsymbol{y};\boldsymbol{\gamma}) \equiv \log|\Sigma_y| + \boldsymbol{y}^T\Sigma_y^{-1}\boldsymbol{y}, \quad (4)$$

where $\Sigma_y \triangleq \lambda I + \Phi\Gamma\Phi^T$ and $\Gamma \triangleq \text{diag}[\boldsymbol{\gamma}]$. Once some $\boldsymbol{\gamma}_* = \arg\min_{\boldsymbol{\gamma}} \mathcal{L}(\boldsymbol{\gamma})$ is computed, an estimate of the unknown coefficients can be obtained by setting $\boldsymbol{x}_{\text{SBL}}$ to the posterior mean computed using $\boldsymbol{\gamma}_*$:

$$\boldsymbol{x}_{\text{SBL}} = \text{E}[\boldsymbol{x}|\boldsymbol{y};\boldsymbol{\gamma}_*] = \Gamma_*\Phi^T\Sigma_{y*}^{-1}\boldsymbol{y}. \quad (5)$$

Note that if any $\gamma_{*,i} = 0$, as often occurs during the learning process, then $x_{\text{SBL},i} = 0$ and the corresponding feature is effectively pruned from the model. The resulting coefficient vector $\boldsymbol{x}_{\text{SBL}}$ is therefore sparse, with nonzero elements corresponding with the 'relevant' features.

It is not immediately apparent how the SBL procedure, which requires optimizing a cost function in $\boldsymbol{\gamma}$-space and is based on a factorial prior $p(\boldsymbol{x};\boldsymbol{\gamma})$, relates to solving/approximating (1) and/or (2) via a non-factorial penalty in $\boldsymbol{x}$-space. However, it has been shown in [19] that $\boldsymbol{x}_{\text{SBL}}$ satisfies

$$\boldsymbol{x}_{\text{SBL}} = \arg\min_{\boldsymbol{x}} \|\boldsymbol{y} - \Phi\boldsymbol{x}\|_2^2 + \lambda g_{\text{SBL}}(\boldsymbol{x}), \quad (6)$$

where

$$g_{\text{SBL}}(\boldsymbol{x}) \triangleq \min_{\boldsymbol{\gamma}\geq 0} \boldsymbol{x}^T\Gamma^{-1}\boldsymbol{x} + \log|\alpha I + \Phi\Gamma\Phi^T|, \quad (7)$$

assuming $\alpha = \lambda$ and $|\boldsymbol{x}| \triangleq [|x_1|, \ldots, |x_m|]^T$. While not discussed in [19], $g_{\text{SBL}}(\boldsymbol{x})$ is a general penalty function that only need have $\alpha = \lambda$ to obtain equivalence with SBL; other selections may lead to better performance (more on this in Section 4 below).

The analysis in [19] reveals that replacing $\|\boldsymbol{x}\|_0$ with $g_{\text{SBL}}(\boldsymbol{x})$ and $\alpha \to 0$ leaves the globally minimizing solution to (1) unchanged but drastically reduces the number of local minima (more so than *any possible* factorial penalty function). While space precludes the details here, these ideas can be extended significantly to form conditions, which again are only satisfiable by a non-factorial penalty, whereby all local minima are smoothed away [21]. Note that while basic $\ell_1$-norm minimization also has no local minima, the global minimum need not always correspond with the global solution to (1), unlike when using $g_{\text{SBL}}(\boldsymbol{x})$.

It can also be shown that $g_{\text{SBL}}(\boldsymbol{x})$ is a non-decreasing, concave function of $|\boldsymbol{x}|$ (see Appendix), a desirable property of sparsity-promoting penalties. Importantly, as a direct consequence of this concavity, (6) can be optimized using a reweighted $\ell_1$ algorithm (in an analogous fashion to the factorial case) using

$$w_i^{(k+1)} = \left.\frac{\partial g_{\text{SBL}}(\boldsymbol{x})}{\partial|x_i|}\right|_{\boldsymbol{x}=\boldsymbol{x}^{(k+1)}}. \quad (8)$$

Although this quantity is not available in closed form (except for the special case where $\alpha \to 0$), it can be estimated by executing: *Step I* - Initialize by setting $\boldsymbol{w}^{(k+1)} \to \boldsymbol{w}^{(k)}$, the $k$-th vector of weights, *Step II* - Repeat until convergence

$$w_i^{(k+1)} \to \left[\boldsymbol{\phi}_i^T \left(\alpha I + \Phi\widetilde{W}^{(k+1)}\widetilde{X}^{(k+1)}\Phi^T\right)^{-1}\boldsymbol{\phi}_i\right]^{\frac{1}{2}}, \quad (9)$$

where $\widetilde{W}^{(k+1)} \triangleq \text{diag}[\boldsymbol{w}^{(k+1)}]^{-1}$ and $\widetilde{X}^{(k+1)} \triangleq \text{diag}[|\boldsymbol{x}^{(k+1)}|]$. The derivation is shown in the Appendix, while further details and analyses are deferred to [20]. Note that cost function descent is guaranteed with only a single iteration, so we need not execute (9) until convergence. In fact, it can be shown that a more rudimentary form of reweighted $\ell_1$ applied to this model in [19] amounts to performing exactly one such iteration. However, repeated execution of (9) is cheap computationally since it scales as $O\left(nm\|\boldsymbol{x}^{(k+1)}\|_0\right)$, where typically $\|\boldsymbol{x}^{(k+1)}\|_0 \leq n$, and is substantially less intensive than the subsequent $\ell_1$ step given by (3).

From a theoretical standpoint, $\ell_1$ reweighting applied to $g_{\text{SBL}}(\boldsymbol{x})$ is guaranteed to aid performance in the sense described by the following two results, which apply in the case where $\lambda \to 0, \alpha \to 0$. Before proceeding, we define $\text{spark}(\Phi)$ as the smallest number of linearly dependent columns in $\Phi$ [5]. It follows then that $2 \leq \text{spark}(\Phi) \leq n+1$.

**Theorem 1.** When applying iterative reweighted $\ell_1$ using (9) and $w_i^{(1)} \neq 0, \forall i$, the solution sparsity satisfies $\|\boldsymbol{x}^{(k+1)}\|_0 \leq \|\boldsymbol{x}^{(k)}\|_0$ (i.e., continued iteration can never do worse).

**Theorem 2.** Assume that $\mathrm{spark}(\Phi) = n+1$ and consider any instance where standard $\ell_1$ minimization fails to find some $\boldsymbol{x}^*$ drawn from support set $\mathcal{S}$ with cardinality $|\mathcal{S}| < \frac{(n+1)}{2}$. Then there exists a set of signals $\boldsymbol{y}$ (with non-zero measure) generated from $\mathcal{S}$ such that non-factorial reweighted $\ell_1$, with $\widetilde{W}^{(k+1)}$ updated using (9), always succeeds but standard $\ell_1$ always fails.

Note that Theorem 2 does not in any way indicate what is the best non-factorial reweighting scheme in practice (for example, in our limited experience with empirical simulations, the selection $\alpha \to 0$ is not necessarily always optimal). However, it does suggest that reweighting with non-convex, non-factorial penalties is potentially very effective, motivating other selections as discussed next. Taken together, Theorems 1 and 2 challenge the prevailing reliance on strictly convex cost functions, since they ensure that we can never do worse than the Lasso (which uses the tightest convex approximation to the $\ell_0$ norm), and that there will always be cases where improvement over the Lasso is obtained.

## 3 Bottom-Up Construction of Non-Factorial Penalty

In the previous section, we described what amounts to a top-down formulation of a non-factorial penalty function that emerges from a particular hierarchical Bayesian model. Based on the insights gleaned from this procedure (and its distinction from factorial penalties), it is possible to stipulate alternative penalty functions from the bottom up by creating plausible, non-factorial reweighting schemes. The following is one such possibility.

Assume for simplicity that $\lambda \to 0$. The Achilles heel of standard, factorial penalties is that if we want to retain a global minimum similar to that of (1), we require a highly concave penalty on each $x_i$ [21]. However, this implies that almost all *basic feasible solutions* (BFS) to $\boldsymbol{y} = \Phi\boldsymbol{x}$, defined as a solution with $\|\boldsymbol{x}\|_0 \leq n$, will form local minima of the penalty function constrained to the feasible region. This is a very undesirable property since there are on the order of $\binom{m}{n}$ BFS with $\|\boldsymbol{x}\|_0 = n$, which is equal to the signal dimension and not very sparse. We would really like to find *degenerate* BFS, where $\|\boldsymbol{x}\|_0$ is strictly less than $n$. Such solutions are exceedingly rare and difficult to find. Consequently we would like to utilize a non-factorial, yet highly concave penalty that explicitly favors degenerate BFS. We can accomplish this by constructing a reweighting scheme designed to avoid non-degenerate BFS whenever possible.

Now consider the covariance-like quantity $\alpha I + \Phi(\widetilde{X}^{(k+1)})^2 \Phi^T$, where $\alpha$ may be small, and then construct weights using the projection of each basis vector $\boldsymbol{\phi}_i$ as defined via

$$w_i^{(k+1)} \to \boldsymbol{\phi}_i^T \left( \alpha I + \Phi(\widetilde{X}^{(k+1)})^2 \Phi^T \right)^{-1} \boldsymbol{\phi}_i. \tag{10}$$

Ideally, if at iteration $k+1$ we are at a bad or non-degenerate BFS, we do not want the newly computed $w_i^{(k+1)}$ to favor the present position at the next iteration of (3) by assigning overly large weights to the zero-valued $x_i$. In such a situation, the factor $\Phi(\widetilde{X}^{(k+1)})^2 \Phi^T$ in (10) will be full rank and so all weights will be relatively modest sized. In contrast, if a rare, degenerate BFS is found, then $\Phi(\widetilde{X}^{(k+1)})^2 \Phi^T$ will no longer be full rank, and the weights associated with zero-valued coefficients will be set to large values, meaning this solution will be favored in the next iteration.

In some sense, the distinction between (10) and its factorial counterparts, such as the method of Candès et al. [3] which uses $w_i^{(k+1)} \to 1/(|x_i^{(k+1)}|+\alpha)$, can be summarized as follows: the factorial methods assign the largest weight *whenever* the associated coefficient goes to zero; with (10) the largest weight is only assigned when the associated coefficient goes to zero *and* $\|\boldsymbol{x}^{(k+1)}\|_0 < n$.

The reweighting option (10), which bears some resemblance to (9), also has some very desirable properties beyond the intuitive justification given above. First, since we are utilizing (10) in the context of reweighted $\ell_1$ minimization, it would productive to know what cost function, if any, we are minimizing when we compute each iteration. Using the fundamental theorem of calculus for line integrals (or the gradient theorem), it follows that the bottom-up (BU) penalty function associated

with (10) is

$$g_{\text{BU}}(\boldsymbol{x}) \triangleq \int_0^1 \text{trace}\left[\widetilde{X}\Phi^T\left(\alpha I + \Phi(\nu\widetilde{X})^2\Phi^T\right)^{-1}\Phi\right]d\nu. \tag{11}$$

Moreover, because each weight $w_i$ is a non-increasing function of each $x_j, \forall j$, from Kachurovskii's theorem [12] it directly follows that (11) is concave and non-decreasing in $|\boldsymbol{x}|$, and thus naturally promotes sparsity. Additionally, for $\alpha$ sufficiently small, it can be shown that the global minimum of (11) on the constraint $\boldsymbol{y} = \Phi\boldsymbol{x}$ must occur at a degenerate BFS (Theorem 1 from above also holds when using (10); Theorem 2 may as well, although we have not formally shown this). And finally, regarding implementational issues and interpretability, (10) avoids any recursive weight assignments or inner-loop optimization as when using (9).

## 4 Extensions

One of the motivating factors for using iterative reweighted $\ell_1$ optimization is that it is very easy to incorporate alternative likelihoods and priors. This section addresses three such examples.

***Non-Negative Sparse Coding***: Numerous applications require sparse solutions where all coefficients $x_i$ are constrained to be non-negative [2]. By adding the contraint $\boldsymbol{x} \geq 0$ to (3) at each iteration, we can easily compute such solutions using $g_{\text{SBL}}(\boldsymbol{x}), g_{\text{BU}}(\boldsymbol{x})$, or any other appropriate penalty function. Note that in the original SBL formulation, this is not a possibility since the integrals required to compute the associated cost function or update rules no longer have closed-form expressions.

***Group Feature Selection***: Another common generalization is to seek sparsity at the level of groups of features, e.g., the group Lasso [23]. The simultaneous sparse approximation problem [17] is a particularly useful adaptation of this idea relevant to compressive sensing [18], manifold learning [13], and neuroimaging [22]. In this situation, we are presented with $r$ signals $Y \triangleq [\boldsymbol{y}_{\cdot 1}, \boldsymbol{y}_{\cdot 2}, \dots, \boldsymbol{y}_{\cdot r}]$ that were produced by coefficient vectors $X \triangleq [\boldsymbol{x}_{\cdot 1}, \boldsymbol{x}_{\cdot 2}, \dots, \boldsymbol{x}_{\cdot r}]$ characterized by the same sparsity profile or support, meaning that the coefficient matrix $X$ is *row sparse*. Here we adopt the notation that $\boldsymbol{x}_{\cdot j}$ represents the $j$-th column of $X$ while $\boldsymbol{x}_{i\cdot}$ represents the $i$-th row of $X$. The sparse recovery problems (1) and (2) then become

$$\min_X d(X), \text{ s.t. } Y = \Phi X, \qquad \text{and} \qquad \min_X \|Y - \Phi X\|_{\mathcal{F}}^2 + \lambda d(X), \ \lambda > 0, \tag{12}$$

where $d(X) \triangleq \sum_{i=1}^m \mathcal{I}\left[\|\boldsymbol{x}_{i\cdot}\| > 0\right]$ and $\mathcal{I}[\cdot]$ is an indicator function. $d(X)$ favors row sparsity and is a natural extension of the $\ell_0$ norm to the simultaneous approximation problem.

As before, the combinatorial nature of each optimization problem renders them intractable and so approximate procedures are required. All of the algorithms discussed herein can naturally be expanded to this domain essentially by substituting the scalar coefficient magnitudes from a given iteration $|x_i^{(k)}|$ with some row-vector penalty, such as a norm. If we utilize $\|\boldsymbol{x}_{i\cdot}\|_2$, then the coefficient matrix update analogous to (3) requires the solution of the more complicated weighted second-order cone (SOC) program

$$X^{(k+1)} \rightarrow \arg\min_X \|Y - \Phi X\|_{\mathcal{F}}^2 + \lambda \sum_i w_i^{(k)}\|\boldsymbol{x}_{i\cdot}\|_2. \tag{13}$$

Other selections such as the $\ell_\infty$ norm are possible as well, providing added generality.

***Sparse Classifier Design***: At a high level, sparse classifiers can be trained by simply substituting a (preferrably) convex likelihood function for the quadratic term in (2). For example, to perform sparse logistic regression we would solve

$$\min_{\boldsymbol{x}} \sum_j \left[y_j \log(\boldsymbol{\phi}_{j\cdot}^T \boldsymbol{x}) + (1 - y_j)\log(1 - \boldsymbol{\phi}_{j\cdot}^T \boldsymbol{x})\right] + \lambda g(\boldsymbol{x}), \tag{14}$$

where now $y_j \in \{0, 1\}$ and $g(\boldsymbol{x})$ is an arbitrary, concave-in-$|\boldsymbol{x}|$ penalty. This can be implemented by iteratively solving an $\ell_1$-norm penalized logistic regression problem, which can be efficiently accomplished using a simple majorization-maximization approach [7]. Note that cost function descent does not require that we compute the full reweighted $\ell_1$ solution; the iterations from [7] naturally lend themselves to an efficient partial (or greedy) update before recomputing the weights.

It is very insightful to compare this methodology with the original SBL (or relevance vector machine) classifier derived in [15]. When the Gaussian likelihood $p(\boldsymbol{y}|\boldsymbol{x})$ is replaced with a Bernoulli

distribution (which leads to the logistic data fit term above), it is no longer possible to compute the marginalization (4) or the posterior distribution $p(\boldsymbol{x}|\boldsymbol{y};\boldsymbol{\gamma})$, which is used both for optimization purposes and to make predictive statements about test data. Consequently, a heuristic Laplace approximation is adopted, which requires a second-order Newton inner-loop to fit a Gaussian about the mode of $p(\boldsymbol{x}|\boldsymbol{y};\boldsymbol{\gamma})$. This Gaussian is then used to transform the classification problem into a standard regression one with data-dependent (herteroscedastic) noise, and then whatever approach is used to minimize (4), either the MacKay update rules [15] or a greedy constructive method [16], can be used in the outer-loop. When (if) a fixed point $\boldsymbol{\gamma}_*$ is reached, the corresponding classifier coefficients are chosen as the mode of $p(\boldsymbol{x}|\boldsymbol{y};\boldsymbol{\gamma}_*)$.

While demonstrably effective in a wide variety of empirical classification tests, the problem with this formulation of SBL is threefold. First, there are no convergence guarantees of any kind, regardless of which method is used for the outer-loop. Secondly, it is completely unclear what, if any, cost function is being descended (even approximately) to obtain the classifier coefficients, making it difficult to explore the model for enhancements or analytical purposes. Thirdly, in certain applications it has been observed that SBL achieves extreme sparsity at the expense of classification accuracy [4, 11]. There is currently no flexibility in the model to remedy this problem.

These issues are directly addressed by dispensing with the Bayesian hierarchical derivation of SBL altogether and considering classification in light of (14). Both the MacKay and greedy SBL updates are equivalent to minimizing (14) with $g(\boldsymbol{x}) = g_{\mathrm{SBL}}(\boldsymbol{x})$, and assuming $\alpha = \lambda = 1$, using coordinate descent over a set of auxiliary functions (details provided in a forthcoming paper). Unfortunately however, because these auxiliary functions are based in part on a second-order Laplace approximation, they do not form a strict upper bound and so provable convergence (or even descent) is not possible. Of course we can always substitute the reweighted $\ell_1$ scheme discussed above to avoid this issue, since the underlying cost function in $\boldsymbol{x}$-space is the same. Perhaps more importantly, to properly regulate sparsity, when we deviate from the original Bayesian inspiration for this model, we are free to adjust $\alpha$ and/or $\lambda$. For example, with $\alpha$ small, the penalty $g_{\mathrm{SBL}}(\boldsymbol{x})$ is more highly concave favoring sparsity, while in the limit at $\alpha$ becomes large, it acts like a standard $\ell_1$ norm, still favoring sparsity but not exceedingly so (the same phenomena occurs when using the penalty (11)). Likewise, $\lambda$ is as a natural trade-off parameter balancing the contribution from the two terms in (6) or (14). Both $\alpha$ and $\lambda$ can be tuned via cross-validation if desired.

There is one additional concern regarding SBL that involves marginal likelihood (sometimes called *evidence*) calculations. In the standard regression case where marginalization was possible, the optimized quantity $-\log p(\boldsymbol{y};\boldsymbol{\gamma})$ represents an approximation to $-\log p(\boldsymbol{y})$ that can be used, among other things, for model comparison. This notion is completely lost when we move to the classification case under consideration. While space precludes the details, if we are willing to substitute a probit likelihood function for the logistic, it is possible to revert (14) back to the original hierarchical, $\boldsymbol{\gamma}$-dependent Bayesian model and obtain a rigorous upper bound on $-\log p(\boldsymbol{y};\boldsymbol{\gamma})$. Finally, detailed empirical simulations with both logistic- and probit-based classifiers is an area of future research; preliminary results are promising.

## 5  Empirical Comparisons

To further examine the algorithms discussed herein, we performed simulations similar to those in [3]. In the first experiment, each trial consisted of generating a $100 \times 256$ dictionary $\Phi$ with iid Gaussian entries and a sparse vector $\boldsymbol{x}^*$ with 60 nonzero, non-negative (truncated Gaussian) coefficients. A signal is then computed using $\boldsymbol{y} = \Phi\boldsymbol{x}^*$. We then attempted to recover $\boldsymbol{x}^*$ by applying nonnegative $\ell_1$ reweighting strategies with four different penalty functions: (i) $g_{\mathrm{SBL}}(\boldsymbol{x})$ implemented using a single iteration of (9), referred to as SBL-I (equivalent to the method from [19]); (ii) $g_{\mathrm{SBL}}(\boldsymbol{x})$ implemented using multiple iterations of (9) as discussed in Section 2, referred to as SBL-II; (iii) $g_{\mathrm{BU}}(\boldsymbol{x})$; and finally (iv) $g(\boldsymbol{x}) = \sum_i \log(|x_i| + \alpha)$, the factorial method of Candès et al., which represents the current state-of-the-art in reweighted $\ell_1$ algorithms. In all cases $\alpha$ was chosen via coarse cross-validation. Additionally, since we are working with a noise-free signal, we assume $\lambda \to 0$ and so the requisite coefficient update (3) with $x_i \geq 0$ reduces to a standard linear program.

Given $w_i^{(0)} = 1, \forall i$ for each algorithm, the first iteration amounts to the non-negative minimum $\ell_1$-norm solution (i.e., the Lasso). Average results from 1000 random trials are displayed in Figure 1 (*left*), which plots the empirical probability of success in recovering $\boldsymbol{x}^*$ versus the iteration number. We observe that standard non-negative $\ell_1$ never succeeds (see first iteration results); however,

with only a few reweighted iterations drastic improvement is possible, especially for the bottom-up approach. By 10 iterations, the non-factorial variants have all exceeded the method of Candès et al. (There was no appreciable improvement by any method after 10 iterations.) This shows both the efficacy of non-factorial reweighting and the ability to handle constraints on $\boldsymbol{x}$.

For the second experiment, we used a randomly generated $50 \times 100$ dictionary for each trial with iid Gaussian entries as above, and created 5 coefficient vectors $X^* = [\boldsymbol{x}_{\cdot 1}^*, ..., \boldsymbol{x}_{\cdot 5}^*]$ with matching sparsity profile and iid Gaussian nonzero coefficients. We then generate the signal matrix $Y = \Phi X^*$ and attempt to learn $X^*$ using various group-level reweighting schemes. In this experiment we varied the row sparsity of $X^*$ from $d(X^*) = 30$ to $d(X^*) = 40$; in general, the more nonzero rows, the harder the recovery problem becomes. A total of five algorithms modified to the simultaneous sparse approximation problem were tested using an $\ell_2$-norm penalty on each coefficient row: the four methods from above (executed for 5 iterations each) plus the standard group Lasso (equivalent to a single iteration of any of the other algorithms). Results are presented in Figure 1 (*right*), where the performance gap between the factorial and non-factorial approaches is very significant. Additionally, we have successfully applied this methodology to large neuroimaging data sets [22], obtaining significant improvements over existing convex approaches such as the group Lasso, consistent with the results in Figure 1. Other related simulation results are contained in [20].

row sparsity, $d(X^*)$

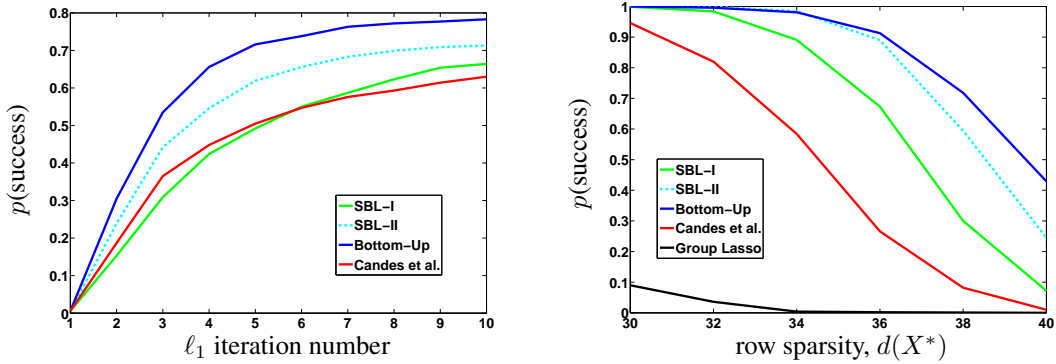

Figure 1: *Left*: Probability of success recovering sparse *non-negative* coefficients as a function of reweighted $\ell_1$ iterations. *Right*: Iterative reweighted results using 5 simultaneous signal vectors. Probability of success recovering sparse coefficients for different row sparsity values, i.e., $d(X^*)$.

## 6   Conclusion

In this paper we have examined concave, non-factorial priors (which previously have received little attention) for the purpose of estimating sparse coefficients. When coupled with general likelihood models and minimized using efficient iterative reweighted $\ell_1$ methods, these priors offer a powerful alternative to existing state-of-the-art sparse estimation techniques. We have also shown (for the first time) exactly what the underlying cost function associated with the SBL classifier is and provided a more principled algorithm for minimizing it.

## Appendix

***Concavity of $g_{SBL}(\boldsymbol{x})$ and derivation of weight updates (9)***: Because $\log|\alpha I + \Phi\Gamma\Phi^T|$ is concave and non-decreasing with respect to $\boldsymbol{\gamma} \geq 0$, we can express it as

$$\log|\alpha I + \Phi\Gamma\Phi^T| = \min_{\boldsymbol{z} \geq 0} \boldsymbol{z}^T\boldsymbol{\gamma} - h^*(\boldsymbol{z}), \qquad (15)$$

where $h^*(\boldsymbol{z})$ is defined as the concave conjugate of $h(\boldsymbol{\gamma}) \triangleq \log|\alpha I + \Phi\Gamma\Phi^T|$ [1]. We can then express $g_{SBL}(\boldsymbol{x})$ via

$$g_{SBL}(\boldsymbol{x}) = \min_{\boldsymbol{\gamma} \geq 0} \boldsymbol{x}^T\Gamma^{-1}\boldsymbol{x} + \log|\alpha I + \Phi\Gamma\Phi^T| = \min_{\boldsymbol{\gamma},\boldsymbol{z} \geq 0} \sum_i \left(\frac{x_i^2}{\gamma_i} + z_i\gamma_i\right) - h^*(\boldsymbol{z}). \qquad (16)$$

Minimizing over $\boldsymbol{\gamma}$ for fixed $\boldsymbol{x}$ and $\boldsymbol{z}$, we get

$$\gamma_i = z_i^{-1/2}|x_i|, \forall i. \qquad (17)$$

Substituting this expression into (16) gives the representation

$$g_{\text{SBL}}(\boldsymbol{x}) = \min_{\boldsymbol{z} \geq 0} \sum_i \left( \frac{x_i^2}{z_i^{-1/2}|x_i|} + z_i z_i^{-1/2}|x_i| \right) - h^*(\boldsymbol{z}) = \min_{\boldsymbol{z} \geq 0} \sum_i 2 z_i^{1/2}|x_i| - h^*(\boldsymbol{z}), \quad (18)$$

which implies that $g_{\text{SBL}}(\boldsymbol{x})$ can be represented as a minimum of upper-bounding hyperplanes with respect to $|\boldsymbol{x}|$, and thus must be concave and non-decreasing since $\boldsymbol{z} \geq 0$ [1]. We also observe that for fixed $\boldsymbol{z}$, solving (6) is a weighted $\ell_1$ minimization problem.

To derive the weight update (9), we only need the optimal value of each $z_i$, which from basic convex analysis will satisfy

$$z_i^{1/2} = \frac{\partial g_{\text{SBL}}(\boldsymbol{x})}{2 \partial |x_i|}. \tag{19}$$

Since this quantity is not available in closed form, we can instead iteratively minimize (16) over $\boldsymbol{\gamma}$ and $\boldsymbol{z}$. We start by initializing $z_i^{1/2} \to w_i^{(k)}, \forall i$ and then minimize over $\boldsymbol{\gamma}$ using (17). We then compute the optimal $\boldsymbol{z}$ for fixed $\boldsymbol{\gamma}$, which can be done analytically using

$$\boldsymbol{z} = \nabla_{\boldsymbol{\gamma}} \log |\alpha I + \Phi \Gamma \Phi^T| = \text{diag}\left[ \Phi^T \left( \alpha I + \Phi \Gamma \Phi^T \right)^{-1} \Phi \right]. \tag{20}$$

By substituting (17) into (20) and defining $w_i^{(k+1)} \triangleq z_i^{1/2}$, we obtain the weight update (9). This procedure is guaranteed to converge to a solution satisfying (19) [20] although, as mentioned previously, only one iteration is actually required for the overall algorithm. ∎

***Proof of Theorem 1***: Before we begin, we should point out that for $\alpha \to 0$, the weight update (9) is still well-specified regardless of the value of the diagonal matrix $\widetilde{W}^{(k+1)} \widetilde{X}^{(k+1)}$. If $\boldsymbol{\phi}_i$ is not in the span of $\Phi \widetilde{W}^{(k+1)} \widetilde{X}^{(k+1)} \Phi^T$, then $w_i^{(k+1)} \to \infty$ and the corresponding coefficient $x_i$ can be set to zero for all future iterations. Otherwise $w_i^{(k+1)}$ can be computed efficiently using the Moore-Penrose pseudoinverse and will be strictly nonzero.

For simplicity we will now assume that $\text{spark}(\Phi) = n + 1$, which is equivalent to requiring that each subset of $n$ columns of $\Phi$ forms a basis in $\mathbb{R}^n$. The extension to the more general case is discussed in [20]. From basic linear programming [8], at any iteration the coefficients will satisfy $\|\boldsymbol{x}^{(k)}\|_0 \leq n$ for arbitrary weights $\widetilde{W}^{(k-1)}$. Given our simplifying assumptions, there exists only two possibilities. If $\|\boldsymbol{x}^{(k)}\|_0 = n$, then we will automatically satisfy $\|\boldsymbol{x}^{(k+1)}\|_0 \leq \|\boldsymbol{x}^{(k)}\|_0$ at the next iteration regardless of $\widetilde{W}^{(k)}$. In contrast, if $\|\boldsymbol{x}^{(k)}\|_0 < n$, then $\text{rank}\left[ \widetilde{W}^{(k)} \right] \leq \|\boldsymbol{x}^{(k)}\|_0$ for all evaluations of (9) with $\alpha \to 0$, enforcing $\|\boldsymbol{x}^{(k+1)}\|_0 \leq \|\boldsymbol{x}^{(k)}\|_0$. ∎

***Proof of Theorem 2***: For a fixed dictionary $\Phi$ and coefficient vector $\boldsymbol{x}^*$, we are assuming that $\|\boldsymbol{x}^*\|_0 < \frac{(n+1)}{2}$. Now consider a second coefficient vector $\boldsymbol{x}'$ with support and sign pattern equal to $\boldsymbol{x}^*$ and define $x'_{(i)}$ as the $i$-th largest coefficient magnitude of $\boldsymbol{x}'$. Then there exists a set of $\|\boldsymbol{x}^*\|_0 - 1$ scaling constants $\nu_i \in (0, 1]$ (i.e., strictly greater than zero) such that, for any signal $\boldsymbol{y}$ generated via $\boldsymbol{y} = \Phi \boldsymbol{x}'$ and $x'_{(i+1)} \leq \nu_i x'_{(i)}, i = 1, \ldots, \|\boldsymbol{x}^*\|_0 - 1$, the minimization problem

$$\hat{\boldsymbol{x}} \triangleq \arg\min_{\boldsymbol{x}} g_{\text{SBL}}(\boldsymbol{x}), \quad \text{s.t. } \Phi \boldsymbol{x}' = \Phi \boldsymbol{x}, \alpha \to 0, \tag{21}$$

is unimodal and has a unique minimizing stationary point which satisfies $\hat{\boldsymbol{x}} = \boldsymbol{x}'$. This result follows from [21] and the dual-space characterization of the penalty $g_{\text{SBL}}(\boldsymbol{x})$ from [19]. Note that (21) is equivalent to (6) with $\lambda \to 0$, so the reweighted non-factorial update (9) can be applied. Furthermore, based on the global convergence of these updates discussed above, the sequence of estimates are guaranteed to satisfy $\boldsymbol{x}^{(k)} \to \hat{\boldsymbol{x}} = \boldsymbol{x}'$. So we will necessarily learn the generative $\boldsymbol{x}'$.

Let $\boldsymbol{x}^{\ell_1} \triangleq \arg\min_{\boldsymbol{x}} \|\boldsymbol{x}\|_1$, subject to $\Phi \boldsymbol{x}^* = \Phi \boldsymbol{x}$. By assumption we know that $\boldsymbol{x}^{\ell_1} \neq \boldsymbol{x}^*$. Moreover, we can conclude using [9, Theorem 6] that if $\boldsymbol{x}^{\ell_1}$ fails for some $\boldsymbol{x}^*$, it will fail for any other $\boldsymbol{x}$ with matching support and sign pattern; it will therefore fail for any $\boldsymbol{x}'$ as defined above. Finally, by construction, the set of feasible $\boldsymbol{x}'$ will have nonzero measure over the support $\mathcal{S}$ since each $\nu_i$ is strictly nonzero. Note also that this result can likely be extended to the case where $\text{spark}(\Phi) < n + 1$ and to any $\boldsymbol{x}^*$ that satisfies $\|\boldsymbol{x}^*\|_0 < \text{spark}(\Phi) - 1$. The more specific case addressed above was only assumed to allow direct application of [9, Theorem 6]. ∎

# References

[1] S. Boyd and L. Vandenberghe, *Convex Optimization*, Cambridge University Press, 2004.

[2] A. Bruckstein, M. Elad, and M. Zibulevsky, "A non-negative and sparse enough solution of an underdetermined linear system of equations is unique," *IEEE Trans. Information Theory*, vol. 54, no. 11, pp. 4813–4820, Nov. 2008.

[3] E. Candès, M. Wakin, and S. Boyd, "Enhancing sparsity by reweighted $\ell_1$ minimization," *J. Fourier Anal. Appl.*, vol. 14, no. 5, pp. 877–905, 2008.

[4] G. Cawley and N. Talbot, "Gene selection in cancer classification using sparse logistic regression with Bayesian regularization," *Bioinformatics*, vol. 22, no. 19, pp. 2348–2355, 2006.

[5] D. Donoho and M. Elad, "Optimally sparse representation in general (nonorthogonal) dictionaries via $\ell_1$ minimization," *Proc. Nat. Acad. Sci.*, vol. 100, no. 5, pp. 2197–2202, 2003.

[6] M. Fazel, H. Hindi, and S. Boyd, "Log-Det heuristic for matrix rank minimization with applications to hankel and Euclidean distance matrices," *Proc. American Control Conf.*, vol. 3, pp. 2156–2162, June 2003.

[7] B. Krishnapuram, L. Carin, M. Figueiredo, and A. Hartemink, "Sparse multinomial logistic regression: Fast algorithms and generalization bounds," *IEEE Trans. Pattn Anal. Mach. Intell.*, vol. 27, pp. 957–968, 2005.

[8] D. Luenberger, *Linear and Nonlinear Programming*, Addison–Wesley, Reading, Massachusetts, second edition, 1984.

[9] D. Malioutov, M. Çetin, and A.S. Willsky, "Optimal sparse representations in general overcomplete bases," *IEEE Int. Conf. Acoust., Speech, and Sig. Proc.*, vol. 2, pp. II–793–796, 2004.

[10] R. Neal, *Bayesian Learning for Neural Networks*, Springer-Verlag, New York, 1996.

[11] Y. Qi, T. Minka, R. Picard, and Z. Ghahramani, "Predictive automatic relevance determination by expectation propagation," *Int. Conf. Machine Learning (ICML)*, pp. 85–92, 2004.

[12] R. Showalter, "Monotone operators in Banach space and nonlinear partial differential equations," *Mathematical Surveys and Monographs 49*. AMS, Providence, RI, 1997.

[13] J. Silva, J. Marques, and J. Lemos, "Selecting landmark points for sparse manifold learning," *Advances in Neural Information Processing Systems 18*, pp. 1241–1248, 2006.

[14] R. Tibshirani, "Regression shrinkage and selection via the Lasso," *Journal of the Royal Statistical Society*, vol. 58, no. 1, pp. 267–288, 1996.

[15] M. Tipping, "Sparse bayesian learning and the relevance vector machine," *J. Machine Learning Research*, vol. 1, pp. 211–244, 2001.

[16] M. Tipping and A. Faul, "Fast marginal likelihood maximisation for sparse Bayesian models," *Ninth Int. Workshop. Artificial Intelligence and Statistics*, Jan. 2003.

[17] J. Tropp, "Algorithms for simultaneous sparse approximation. Part II: Convex relaxation," *Signal Processing*, vol. 86, pp. 589–602, April 2006.

[18] M. Wakin, M. Duarte, S. Sarvotham, D. Baron, and R. Baraniuk, "Recovery of jointly sparse signals from a few random projections," *Advances in Neural Information Processing Systems 18*, pp. 1433–1440, 2006.

[19] D. Wipf and S. Nagarajan, "A new view of automatic relevance determination," *Advances in Neural Information Processing Systems 20*, pp. 1625–1632, 2008.

[20] D. Wipf and S. Nagarajan, "Iterative reweighted $\ell_1$ and $\ell_2$ methods for finding sparse solutions," Submitted, 2009.

[21] D. Wipf and S. Nagarajan, "Latent variable Bayesian models for promoting sparsity," Submitted, 2009.

[22] D. Wipf, J. Owen, H. Attias, K. Sekihara, and S. Nagarajan, "Robust Bayesian Estimation of the Location, Orientation, and Time Course of Multiple Correlated Neural Sources using MEG," *NeuroImage*, vol. 49, no. 1, pp. 641–655, Jan. 2010.

[23] M. Yuan and Y. Lin, "Model selection and estimation in regression with grouped variables," *J. R. Statist. Soc. B*, vol. 68, pp. 49–67, 2006.

[24] W. Zangwill, *Nonlinear Programming: A Unified Approach*, Prentice Hall, New Jersey, 1969.
